# Structured Learning of Gaussian Graphical Models

**Karthik Mohan**[*], **Michael Jae-Yoon Chung**[†], **Seungyeop Han**[†],

**Daniela Witten**[‡], **Su-In Lee**[§], **Maryam Fazel**[*]

## Abstract

We consider estimation of multiple high-dimensional Gaussian graphical models corresponding to a single set of nodes under several distinct conditions. We assume that most aspects of the networks are shared, but that there are some structured differences between them. Specifically, the network differences are generated from *node perturbations*: a few nodes are perturbed across networks, and most or all edges stemming from such nodes differ between networks. This corresponds to a simple model for the mechanism underlying many cancers, in which the gene regulatory network is disrupted due to the aberrant activity of a few specific genes. We propose to solve this problem using the *perturbed-node joint graphical lasso*, a convex optimization problem that is based upon the use of a *row-column overlap norm* penalty. We then solve the convex problem using an alternating directions method of multipliers algorithm. Our proposal is illustrated on synthetic data and on an application to brain cancer gene expression data.

## 1   Introduction

Probabilistic graphical models are widely used in a variety of applications, from computer vision to natural language processing to computational biology. As this modeling framework is used in increasingly complex domains, the problem of selecting from among the exponentially large space of possible network structures is of paramount importance. This problem is especially acute in the high-dimensional setting, in which the number of variables or nodes in the graphical model is much larger than the number of observations that are available to estimate it.

As a motivating example, suppose that we have access to gene expression measurements for $n_1$ lung cancer patients and $n_2$ brain cancer patients, and that we would like to estimate the gene regulatory networks underlying these two types of cancer. We can consider estimating a single network on the basis of all $n_1 + n_2$ patients. However, this approach is unlikely to be successful, due to fundamental differences between the true lung cancer and brain cancer gene regulatory networks that stem from tissue specificity of gene expression as well as differing etiology of the two diseases. As an alternative, we could simply estimate a gene regulatory network using the $n_1$ lung cancer patients and a separate gene regulatory network using the $n_2$ brain cancer patients. However, this approach fails to exploit the fact that the two underlying gene regulatory networks likely have substantial commonality, such as tumor-specific pathways. In order to effectively make use of the available data, we need a principled approach for jointly estimating the lung cancer and brain cancer networks in such a way that the two network estimates are encouraged to be quite similar to each other, while allowing for certain structured differences. In fact, these differences themselves may be of scientific interest.

In this paper, we propose a general framework for jointly learning the structure of $K$ networks, under the assumption that the networks are similar overall, but may have certain *structured* differences.

---

[*]Electrical Engineering, Univ. of Washington. {`karna,mfazel`}@uw.edu

[†]Computer Science and Engineering, Univ. of Washington. {`mjyc,syhan`}@cs.washington.edu

[‡]Biostatistics, Univ. of Washington. `dwitten@uw.edu`

[§]Computer Science and Engineering, and Genome Sciences, Univ. of Washington. `suinlee@uw.edu`

Specifically, we assume that the network differences result from *node perturbation* – that is, certain nodes are perturbed across the conditions, and so all or most of the edges associated with those nodes differ across the $K$ networks. We detect such differences through the use of a *row-column overlap norm* penalty. Figure 1 illustrates a toy example in which a pair of networks are identical to each other, except for a single perturbed node ($X_2$) that will be detected using our proposal.

The problem of estimating multiple networks that differ due to node perturbations arises in a number of applications. For instance, the gene regulatory networks in cancer patients and in normal individuals are likely to be similar to each other, with specific node perturbations that arise from a small set of genes with somatic (cancer-specific) mutations. Another example arises in the analysis of the conditional independence relationships among $p$ stocks at two distinct points in time. We might be interested in detecting stocks that have differential connectivity with all other edges across the two time points, as these likely correspond to companies that have undergone significant changes. Still another example can be found in the field of neuroscience, where we are interested in learning how the connectivity of neurons in the human brain changes over time.

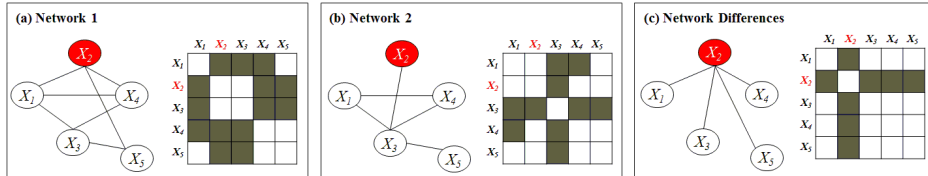

Figure 1: An example of two networks that differ due to *node perturbation* of $X_2$. (a) Network 1 and its adjacency matrix. (b) Network 2 and its adjacency matrix. (c) *Left:* Edges that differ between the two networks. *Right:* Shaded cells indicate edges that differ between Networks 1 and 2.

Our proposal for estimating multiple networks in the presence of node perturbation can be formulated as a convex optimization problem, which we solve using an efficient alternating directions method of multipliers (ADMM) algorithm that significantly outperforms general-purpose optimization tools. We test our method on synthetic data generated from known graphical models, and on one real-world task that involves inferring gene regulatory networks from experimental data.

The rest of this paper is organized as follows. In Section 2, we present recent work in the estimation of Gaussian graphical models (GGMs). In Section 3, we present our proposal for structured learning of multiple GGMs using the row-column overlap norm  penalty. In Section 4, we present an ADMM algorithm that solves the proposed convex optimization problem. Applications to synthetic and real data are in Section 5, and the discussion is in Section 6.

## 2 Background

### 2.1 The graphical lasso

Suppose that we wish to estimate a GGM on the basis of $n$ observations, $X_1, \ldots, X_n \in \mathbb{R}^p$, which are independent and identically distributed $N(\mathbf{0}, \boldsymbol{\Sigma})$. It is well known that this amounts to learning the sparsity structure of $\boldsymbol{\Sigma}^{-1}$ [1, 2]. When $n > p$, one can estimate $\boldsymbol{\Sigma}^{-1}$ by maximum likelihood, but when $p > n$ this is not possible because the empirical covariance matrix is singular. Consequently, a number of authors [3, 4, 5, 6, 7, 8, 9] have considered maximizing the penalized log likelihood

$$\underset{\boldsymbol{\Theta} \in S_{++}^p}{\text{maximize}} \left\{ \log \det \boldsymbol{\Theta} - \text{trace}(\mathbf{S}\boldsymbol{\Theta}) - \lambda \|\boldsymbol{\Theta}\|_1 \right\}, \tag{1}$$

where $\mathbf{S}$ is the empirical covariance matrix based on the $n$ observations, $\lambda$ is a positive tuning parameter, $S_{++}^p$ denotes the set of positive definite matrices of size $p$, and $\|\boldsymbol{\Theta}\|_1$ is the entrywise $\ell_1$ norm. The $\hat{\boldsymbol{\Theta}}$ that solves (1) serves as an estimate of $\boldsymbol{\Sigma}^{-1}$. This estimate will be positive definite for any $\lambda > 0$, and sparse when $\lambda$ is sufficiently large, due to the $\ell_1$ penalty [10] in (1). We refer to (1) as the *graphical lasso formulation*. This formulation is convex, and efficient algorithms for solving it are available [6, 4, 5, 7, 11].

## 2.2 The fused graphical lasso

In recent literature, convex formulations have been proposed for extending the graphical lasso (1) to the setting in which one has access to a number of observations from $K$ distinct conditions. The goal of the formulations is to estimate a graphical model for each condition under the assumption that the $K$ networks share certain characteristics [12, 13]. Suppose that $X_1^k, \ldots, X_{n_k}^k \in \mathbb{R}^p$ are independent and identically distributed from a $N(\mathbf{0}, \mathbf{\Sigma}^k)$ distribution, for $k = 1, \ldots, K$. Letting $\mathbf{S}^k$ denote the empirical covariance matrix for the $k$th class, one can maximize the penalized log likelihood

$$\underset{\mathbf{\Theta}^1 \in S_{++}^p, \ldots, \mathbf{\Theta}^K \in S_{++}^p}{\text{maximize}} \left\{ L(\mathbf{\Theta}^1, \ldots, \mathbf{\Theta}^K) - \lambda_1 \sum_{k=1}^K \|\mathbf{\Theta}^k\|_1 - \lambda_2 \sum_{i \neq j} P(\mathbf{\Theta}_{ij}^1, \ldots, \mathbf{\Theta}_{ij}^K) \right\}, \quad (2)$$

where $L(\mathbf{\Theta}^1, \ldots, \mathbf{\Theta}^K) = \sum_{k=1}^K n_k \left( \log \det \mathbf{\Theta}^k - \text{trace}(\mathbf{S}^k \mathbf{\Theta}^k) \right)$, $\lambda_1$ and $\lambda_2$ are nonnegative tuning parameters, and $P(\mathbf{\Theta}_{ij}^1, \ldots, \mathbf{\Theta}_{ij}^K)$ is a penalty applied to each off-diagonal element of $\mathbf{\Theta}^1, \ldots, \mathbf{\Theta}^K$ in order to encourage similarity among them. Then the $\hat{\mathbf{\Theta}}^1, \ldots, \hat{\mathbf{\Theta}}^K$ that solve (2) serve as estimates for $(\mathbf{\Sigma}^1)^{-1}, \ldots, (\mathbf{\Sigma}^K)^{-1}$. In particular, [13] considered the use of

$$P(\mathbf{\Theta}_{ij}^1, \ldots, \mathbf{\Theta}_{ij}^K) = \sum_{k < k'} |\mathbf{\Theta}_{ij}^k - \mathbf{\Theta}_{ij}^{k'}|, \quad (3)$$

a *fused lasso* penalty [14] on the differences between pairs of network edges. When $\lambda_1$ is large, the network estimates will be sparse, and when $\lambda_2$ is large, pairs of network estimates will have identical edges. We refer to (2) with penalty (3) as the *fused graphical lasso formulation* (FGL).

Solving the FGL formulation allows for much more accurate network inference than simply learning each of the $K$ networks separately, because FGL borrows strength across all available observations in estimating each network. But in doing so, it implicitly assumes that differences among the $K$ networks arise from *edge perturbations*. Therefore, this approach does not take full advantage of the structure of the learning problem, which is that differences between the $K$ networks are driven by *nodes* that differ across networks, rather than differences in individual edges.

# 3 The perturbed-node joint graphical lasso

## 3.1 Why is detecting node perturbation challenging?

At first glance, the problem of detecting node perturbation seems simple: in the case $K = 2$, we could simply modify (2) as follows,

$$\underset{\mathbf{\Theta}^1 \in S_{++}^p, \mathbf{\Theta}^2 \in S_{++}^p}{\text{maximize}} \left\{ L(\mathbf{\Theta}^1, \mathbf{\Theta}^2) - \lambda_1 \|\mathbf{\Theta}^1\|_1 - \lambda_1 \|\mathbf{\Theta}^2\|_1 - \lambda_2 \sum_{j=1}^p \|\mathbf{\Theta}_j^1 - \mathbf{\Theta}_j^2\|_2 \right\}, \quad (4)$$

where $\mathbf{\Theta}_j^k$ is the $j$th column of the matrix $\mathbf{\Theta}^k$. This amounts to applying a *group lasso* [15] penalty to the columns of $\mathbf{\Theta}^1 - \mathbf{\Theta}^2$. Since a group lasso penalty simultaneously shrinks all elements to which it is applied to zero, it appears that this will give the desired node perturbation structure. We will refer to this as the *naive group lasso* approach.

Unfortunately, a problem arises due to the fact that the optimization problem (4) must be performed subject to a symmetry constraint on $\mathbf{\Theta}^1$ and $\mathbf{\Theta}^2$. This symmetry constraint effectively imposes overlap among the elements in the $p$ group lasso penalties in (4), since the $(i, j)$th element of $\mathbf{\Theta}^1 - \mathbf{\Theta}^2$ is in both the $i$th (row) and $j$th (column) groups. In the presence of overlapping groups, the group lasso penalty yields estimates whose *support is the complement of the union of groups* [16, 17]. Figure 2 shows a simple example of $(\mathbf{\Sigma}^1)^{-1} - (\mathbf{\Sigma}^2)^{-1}$ in the case of node perturbation, as well as the estimate obtained using (4). The figure reveals that (4) cannot be used to detect node perturbation, since this task requires a penalty that yields estimates whose *support is the union of groups*.

## 3.2 Proposed approach

A node-perturbation in a GGM can be equivalently represented through a perturbation of the entries of a row and column of the corresponding precision matrix (Figure 1). In other words, we can

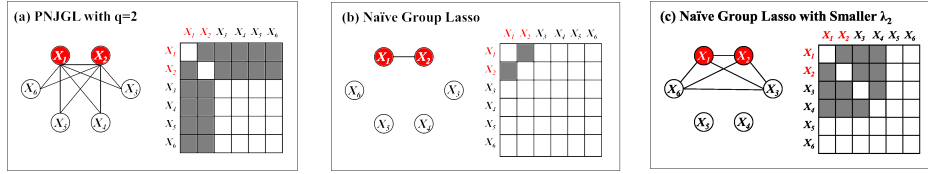

Figure 2: A toy example with $p = 6$ variables, of which two are perturbed (in red). Each panel shows an estimate of $(\mathbf{\Sigma}^1)^{-1} - (\mathbf{\Sigma}^2)^{-1}$, displayed as a network and as an adjacency matrix. Shaded elements of the adjacency matrix indicate non-zero elements of $\hat{\mathbf{\Theta}}^1 - \hat{\mathbf{\Theta}}^2$, as do edges in the network. Results are shown for *(a):* PNJGL with $q = 2$, which gives the correct sparsity pattern; *(b)-(c):* the naive group lasso. The naive group lasso is unable to detect the pattern of node perturbation.

detect a single node perturbation by looking for a row and a corresponding column of $\mathbf{\Theta}^1 - \mathbf{\Theta}^2$ that has nonzero elements. We define a *row-column group* as a group that consists of a row and the corresponding column in a matrix. Note that in a $p \times p$ matrix, there exist $p$ such groups, which overlap. If several nodes of a GGM are perturbed, then this will correspond to the *union of the corresponding row-column groups* in $\mathbf{\Theta}^1 - \mathbf{\Theta}^2$. Therefore, in order to detect node perturbations in a GGM (Figure 1), we must construct a regularizer that can promote estimates whose support is the union of row-column groups. For this task, we propose the *row-column overlap norm* as a penalty.

**Definition 3.1.** *The row-column overlap norm (RCON) induced by a matrix norm $f$ is defined as*

$$\Omega_f(\mathbf{A}) = \min_{\mathbf{V}:\mathbf{A}=\mathbf{V}+\mathbf{V}^T} f(\mathbf{V}). \tag{5}$$

RCON satisfies the following properties that are easy to check: (1) $\Omega_f$ is indeed a norm. Consequently, it is convex. (2) When $f$ is symmetric in its argument, i.e., $f(\mathbf{V}) = f(\mathbf{V}^T)$, then $\Omega_f(\mathbf{A}) = f(\mathbf{A})/2$.

In this paper, we are interested in the particular class of RCON penalty where $f$ is given by

$$f(\mathbf{V}) = \sum_{j=1}^{p} \|\mathbf{V}_j\|_q, \tag{6}$$

where $1 \leq q \leq \infty$. The norm in (6) is known as the $\ell_1/\ell_q$ norm since it can be interpreted as the $\ell_1$ norm of the $\ell_q$ norms of the columns of a matrix. With a little abuse of notation, we will let $\Omega_q$ denote $\Omega_f$ with an $\ell_1/\ell_q$ norm of the form (6). We note that $\Omega_q$ is closely related to the *overlap group lasso penalty* [17, 16], and in fact can be derived from it (for the case of $q = 2$). However, our definition naturally and elegantly handles the grouping structure induced by the overlap of rows and columns, and can accommodate any $\ell_q$ norm with $q \geq 1$, and more generally any norm $f$. As discussed in [17], when applied to $\mathbf{\Theta}^1 - \mathbf{\Theta}^2$, the penalty $\Omega_q$ (with $q = 2$) will encourage the support of the matrix $\hat{\mathbf{\Theta}}^1 - \hat{\mathbf{\Theta}}^2$ to be *the union of a set of rows and columns*.

Now, consider the task of jointly estimating two precision matrices by solving

$$\underset{\mathbf{\Theta}^1 \in S_{++}^p, \mathbf{\Theta}^2 \in S_{++}^p}{\text{maximize}} \left\{ L(\mathbf{\Theta}^1, \mathbf{\Theta}^2) - \lambda_1 \|\mathbf{\Theta}^1\|_1 - \lambda_1 \|\mathbf{\Theta}^2\|_1 - \lambda_2 \Omega_q(\mathbf{\Theta}^1 - \mathbf{\Theta}^2) \right\}. \tag{7}$$

We refer to the convex optimization problem (7) as the *perturbed-node joint graphical lasso* (PN-JGL) formulation. In (7), $\lambda_1$ and $\lambda_2$ are nonnegative tuning parameters, and $q \geq 1$. Note that $f(\mathbf{V}) = \|\mathbf{V}\|_1$ satisfies property 2 of the RCON penalty. Thus we have the following observation.

**Remark 3.1.** *The FGL formulation (2) is a special case of the PNJGL formulation (7) with $q = 1$.*

Let $\hat{\mathbf{\Theta}}^1, \hat{\mathbf{\Theta}}^2$ be the optimal solution to (7). Note that the FGL formulation is an *edge-based* approach that promotes many entries (or edges) in $\hat{\mathbf{\Theta}}^1 - \hat{\mathbf{\Theta}}^2$ to be set to zero. However, setting $q = 2$ or $q = \infty$ in (7) gives us a *node-based approach*, where the support of $\hat{\mathbf{\Theta}}^1 - \hat{\mathbf{\Theta}}^2$ is encouraged to be a union of a few rows and the corresponding columns [17, 16]. Thus the nodes that have been perturbed can be clearly detected using PNJGL with $q = 2, \infty$. An example of the sparsity structure detected by PNJGL with $q = 2$ is shown in the left-hand panel of Figure 2. We note that the above formulation can be easily extended to the estimation of $K > 2$ GGMs by including $\frac{K(K-1)}{2}$ RCON penalty terms in (7), one for each pair of models. However we restrict ourselves to the case of $K = 2$ in this paper.

## 4  An ADMM algorithm for the PNJGL formulation

The PNJGL optimization problem (7) is convex, and so can be directly solved in the modeling environment `cvx` [18], which calls conic interior-point solvers such as `SeDuMi` or `SDPT3`. However, such a general approach does not fully exploit the structure of the problem and will not scale well to large-scale instances. Other algorithms proposed for overlapping group lasso penalties [19, 20, 21] do not apply to our setting since the PNJGL formulation has a combination of Gaussian log-likelihood loss (instead of squared error loss) and the RCON penalty along with a positive-definite constraint. We also note that other first-order methods are not easily applied to solve the PNJGL formulation because the subgradient of the RCON is not easy to compute and in addition the proximal operator to RCON is non-trivial to compute.

In this section we present a fast and scalable *alternating directions method of multipliers* (ADMM) algorithm [22] to solve the problem (7). We first reformulate (7) by introducing new variables, so as to decouple some of the terms in the objective function that are difficult to optimize jointly. This will result in a simple algorithm with closed-form updates. The reformulation is as follows:

$$\underset{\mathbf{\Theta}^1 \in \mathcal{S}_{++}^p, \mathbf{\Theta}^2 \in \mathcal{S}_{++}^p, \mathbf{Z}_1, \mathbf{Z}_2, \mathbf{V}, \mathbf{W}}{\text{minimize}} \left\{ -L(\mathbf{\Theta}^1, \mathbf{\Theta}^2) + \lambda_1 \|\mathbf{Z}_1\|_1 + \lambda_1 \|\mathbf{Z}_2\|_1 + \lambda_2 \sum_{j=1}^{p} \|\mathbf{V}_j\|_q \right\}$$

$$\text{subject to} \quad \mathbf{\Theta}^1 - \mathbf{\Theta}^2 = \mathbf{V} + \mathbf{W}, \mathbf{V} = \mathbf{W}^T, \mathbf{\Theta}^1 = \mathbf{Z}_1, \mathbf{\Theta}^2 = \mathbf{Z}_2. \quad (8)$$

An ADMM algorithm can now be obtained in a standard fashion from the *augmented Lagrangian* to (8). We defer the details to a longer version of this paper. The complete algorithm for (8) is given in Algorithm 1, in which the operator Expand is given by

$$\text{Expand}(\mathbf{A}, \rho, n_k) = \underset{\mathbf{\Theta} \in S_{++}^p}{\text{argmin}} \left\{ -n_k \log \det(\mathbf{\Theta}) + \rho \|\mathbf{\Theta} - \mathbf{A}\|_F^2 \right\} = \frac{1}{2} \mathbf{U} \left( \mathbf{D} + \sqrt{\mathbf{D}^2 + \frac{2n_k}{\rho} \mathbf{I}} \right) \mathbf{U}^T,$$

where $\mathbf{U}\mathbf{D}\mathbf{U}^T$ is the eigenvalue decomposition of $\mathbf{A}$, and as mentioned earlier, $n_k$ is the number of observations in the $k$th class. The operator $\mathcal{T}_q$ is given by

$$\mathcal{T}_q(\mathbf{A}, \lambda) = \underset{\mathbf{X}}{\text{argmin}} \left\{ \frac{1}{2} \|\mathbf{X} - \mathbf{A}\|_F^2 + \lambda \sum_{j=1}^{p} \|\mathbf{X}_j\|_q \right\},$$

and is also known as the proximal operator corresponding to the $\ell_1/\ell_q$ norm. For $q = 1, 2, \infty$, $\mathcal{T}_q$ takes a simple form, which we omit here due to space constraints. A description of these operators can also be found in Section 5 of [25].

Algorithm 1 can be interpreted as an *approximate dual gradient ascent method*. The approximation is due to the fact that the gradient of the dual to the augmented Lagrangian in each iteration is computed inexactly, through a coordinate descent cycling through the primal variables.

Typically ADMM algorithms iterate over only two groups of primal variables. For such algorithms, the convergence properties are well-known (see e.g. [22]). However, in our case we cycle through more than two such groups. Although investigation of the convergence properties of ADMM algorithms for an arbitrary number of groups is an ongoing research area in the optimization literature [23, 24] and specific convergence results for our algorithm are not known, we empirically observe very good convergence behavior. Further study of this issue is a direction for future work.

We initialize the primal variables to the identity matrix, and the dual variables to the matrix of zeros. We set $\mu = 5$, and $t_{\max} = 1000$. In our implementation, the stopping criterion is that the difference between consecutive iterates becomes smaller than a tolerance $\epsilon$. The ADMM algorithm is orders of magnitude faster than an interior point method and also comparable in accuracy. Note that the per-iteration complexity of the ADMM algorithm is $O(p^3)$ (complexity of computing SVD). On the other hand, the complexity of an interior point method is $O(p^6)$. When $p = 30$, the interior point method (using `cvx`, which calls `Sedumi`) takes 7 minutes to run while ADMM takes only 10 seconds. When $p = 50$, the times are 3.5 hours and 2 minutes, respectively. Also, we observe that the average error between the `cvx` and ADMM solution when averaged over many random generations of the data is of $O(10^{-4})$.

**Algorithm 1:** ADMM algorithm for the PNJGL optimization problem (7)

---

**input**: $\rho > 0, \mu > 1, t_{\max} > 0, \epsilon > 0$;

**for** $t = 1{:}t_{\max}$ **do**

    $\rho \leftarrow \mu\rho$ ;

    **while** *Not converged* **do**

        $\mathbf{\Theta}^1 \leftarrow \text{Expand}\left(\frac{1}{2}(\mathbf{\Theta}^2 + \mathbf{V} + \mathbf{W} + \mathbf{Z}_1) - \frac{1}{2\rho}(\mathbf{Q}_1 + n_1 \mathbf{S}_1 + \mathbf{F}), \rho, n_1\right)$ ;

        $\mathbf{\Theta}^2 \leftarrow \text{Expand}\left(\frac{1}{2}(\mathbf{\Theta}^1 - (\mathbf{V} + \mathbf{W}) + \mathbf{Z}_2) - \frac{1}{2\rho}(\mathbf{Q}_2 + n_2 \mathbf{S}_2 - \mathbf{F}), \rho, n_2\right)$ ;

        $\mathbf{Z}_i \leftarrow \mathcal{T}_1\left(\mathbf{\Theta}^i + \frac{\mathbf{Q}_i}{\rho}, \frac{\lambda_1}{\rho}\right)$ for $i = 1, 2$ ;

        $\mathbf{V} \leftarrow \mathcal{T}_q\left(\frac{1}{2}(\mathbf{W}^T - \mathbf{W} + (\mathbf{\Theta}^1 - \mathbf{\Theta}^2)) + \frac{1}{2\rho}(\mathbf{F} - \mathbf{G}), \frac{\lambda_2}{2\rho}\right)$ ;

        $\mathbf{W} \leftarrow \frac{1}{2}(\mathbf{V}^T - \mathbf{V} + (\mathbf{\Theta}^1 - \mathbf{\Theta}^2)) + \frac{1}{2\rho}(\mathbf{F} + \mathbf{G}^T)$ ;

        $\mathbf{F} \leftarrow \mathbf{F} + \rho(\mathbf{\Theta}^1 - \mathbf{\Theta}^2 - (\mathbf{V} + \mathbf{W}))$ ;

        $\mathbf{G} \leftarrow \mathbf{G} + \rho(\mathbf{V} - \mathbf{W}^T)$;

        $\mathbf{Q}_i \leftarrow \mathbf{Q}_i + \rho(\mathbf{\Theta}^i - \mathbf{Z}_i)$ for $i = 1, 2$

---

## 5 Experiments

We describe experiments and report results on both synthetically generated data and real data.

### 5.1 Synthetic experiments

**Synthetic data generation.** We generated two networks as follows. The networks share individual edges as well as hub nodes, or nodes that are highly-connected to many other nodes. There are also perturbed nodes that differ between the networks. We first create a $p \times p$ symmetric matrix $\mathbf{A}$, with diagonal elements equal to one. For $i < j$, we set

$$A_{ij} \sim_{i.i.d.} \begin{cases} 0 & \text{with probability } 0.98 \\ \text{Unif}([-0.6, -0.3] \cup [0.3, 0.6]) & \text{otherwise} \end{cases},$$

and then we set $A_{ji}$ to equal $A_{ij}$. Next, we randomly selected seven *hub nodes*, and set the elements of the corresponding rows and columns to be i.i.d. from a $\text{Unif}([-0.6, -0.3] \cup [0.3, 0.6])$ distribution. These steps resulted in a background pattern of structure common to both networks. Next, we copied $\mathbf{A}$ into two matrices, $\mathbf{A}^1$ and $\mathbf{A}^2$. We randomly selected $m$ *perturbed nodes* that differ between $\mathbf{A}^1$ and $\mathbf{A}^2$, and set the elements of the corresponding row and column of either $\mathbf{A}^1$ or $\mathbf{A}^2$ (chosen at random) to be i.i.d. draws from a $\text{Unif}([-1.0, -0.5] \cup [0.5, 1.0])$ distribution. Finally, we computed $c = \min(\lambda_{min}(\mathbf{A}^1), \lambda_{min}(\mathbf{A}^2))$, the smallest eigenvalue of $\mathbf{A}^1$ and $\mathbf{A}^2$. We then set $(\mathbf{\Sigma}^1)^{-1}$ equal to $\mathbf{A}^1 + (0.1 - c)\mathbf{I}$ and set $(\mathbf{\Sigma}^2)^{-1}$ equal to $\mathbf{A}^2 + (0.1 - c)\mathbf{I}$. This last step is performed in order to ensure positive definiteness. We generated $n$ independent observations each from a $N(\mathbf{0}, \mathbf{\Sigma}^1)$ and a $N(\mathbf{0}, \mathbf{\Sigma}^2)$ distribution, and used these to compute the empirical covariance matrices $\mathbf{S}^1$ and $\mathbf{S}^2$. We compared the performances of graphical lasso, FGL, and PNJGL with $q = 2$ with $p = 100$, $m = 2$, and $n = \{10, 25, 50, 200\}$.

**Results.** Results (averaged over 100 iterations) are shown in Figure 3. Increasing $n$ yields more accurate results for PNJGL with $q = 2$, FGL, and graphical lasso. Furthermore, PNJGL with $q = 2$ identifies non-zero edges and differing edges much more accurately than does FGL, which is in turn more accurate than graphical lasso. PNJGL also leads to the most accurate estimates of $\mathbf{\Theta}^1$ and $\mathbf{\Theta}^2$. The extent to which PNJGL with $q = 2$ outperforms others is more apparent when $n$ is small.

### 5.2 Inferring biological networks

We applied the PNJGL method to a recently-published cancer gene expression data set [26], with mRNA expression measurements for 11,861 genes in 220 patients with glioblastoma multiforme (GBM), a brain cancer. Each patient has one of four distinct clinical subtypes: Proneural, Neural, Classical, and Mesenchymal. We selected two subtypes – Proneural (53 patients) and Mesenchymal

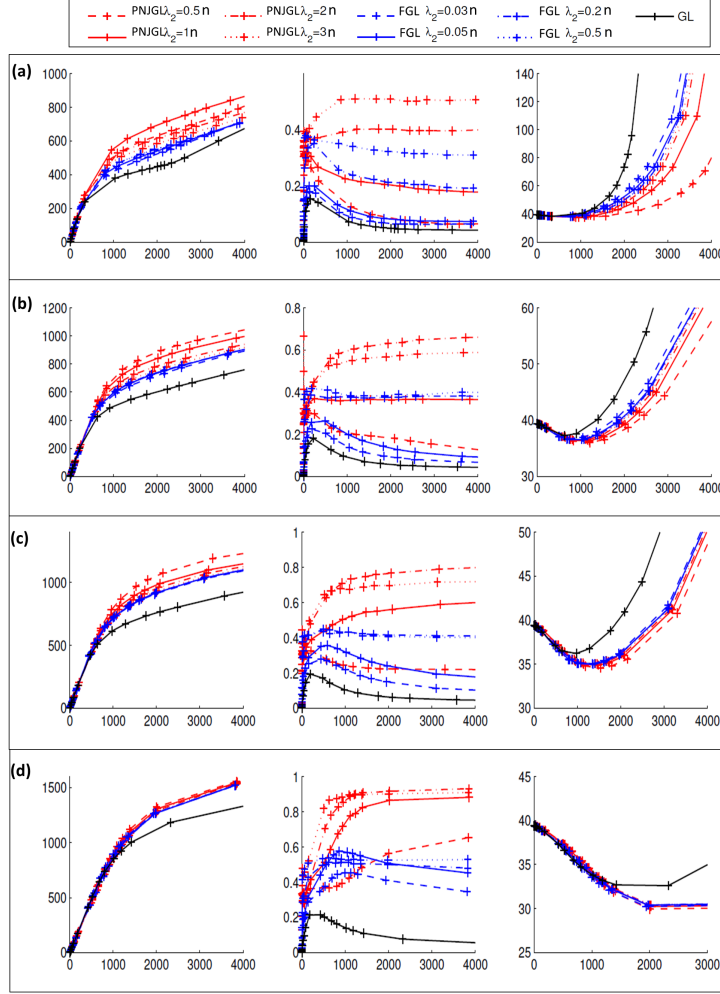

Figure 3: Simulation study results for PNJGL with $q = 2$, FGL, and the graphical lasso (GL), for (a) $n = 10$, (b) $n = 25$, (c) $n = 50$, (d) $n = 200$, when $p = 100$. Within each panel, each line corresponds to a fixed value of $\lambda_2$ (for PNJGL with $q = 2$ and for FGL). Each plot's $x$-axis denotes the number of edges estimated to be non-zero. The $y$-axes are as follows. *Left:* Number of edges *correctly* estimated to be non-zero. *Center:* Number of edges *correctly* estimated to differ across networks, *divided by* the number of edges estimated to differ across networks. *Right:* The Frobenius norm of the error in the estimated precision matrices, i.e. $\left(\sum_{i \neq j}(\theta_{ij}^1 - \hat{\theta}_{ij}^1)^2\right)^{1/2} + \left(\sum_{i \neq j}(\theta_{ij}^2 - \hat{\theta}_{ij}^2)^2\right)^{1/2}$.

(56 patients) – for our analysis. In this experiment, we aim to reconstruct the gene regulatory networks of the two subtypes, as well as to identify genes whose interactions with other genes vary significantly between the subtypes. Such genes are likely to have many *somatic* (cancer-specific) mutations. Understanding the molecular basis of these subtypes will lead to better understanding of brain cancer, and eventually, improved patient treatment. We selected the 250 genes with the highest within-subtype variance, as well as 10 genes known to be frequently mutated across the four GBM subtypes [26]: TP53, PTEN, NF1, EGFR, IDH1, PIK3R1, RB1, ERBB2, PIK3CA, PDGFRA. Two of these genes (EGFR, PDGFRA) were in the initial list of 250 genes selected based on the within-subtype variance. This led to a total of 258 genes. We then applied PNJGL with $q = 2$ and FGL to the resulting $53 \times 258$ and $56 \times 258$ gene expression datasets, after standardizing each gene to have variance one. Tuning parameters were selected so that each approach results in a per-network estimate of approximately 6,000 non-zero edges, as well as approximately 4,000 edges that differ

across the two network estimates. However, the results that follow persisted across a wide range of tuning parameter values.

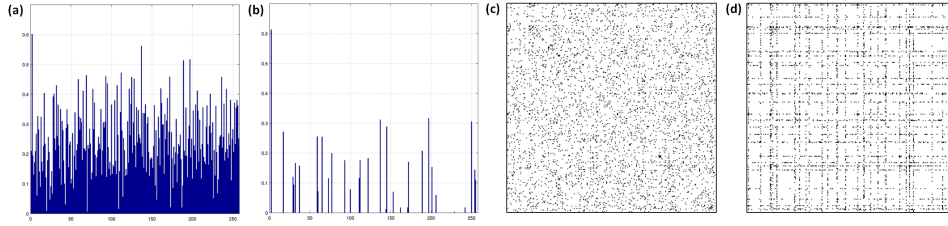

Figure 4: PNJGL with $q = 2$ and FGL were performed on the brain cancer data set corresponding to 258 genes in patients with Proneural and Mesenchymal subtypes. (a)-(b): $NP_j$ is plotted for each gene, based on (a) the FGL estimates and (b) the PNJGL estimates. (c)-(d): A heatmap of $\hat{\Theta}^1 - \hat{\Theta}^2$ is shown for (c) FGL and (d) PNJGL; zero values are in white, and non-zero values are in black.

We quantify the extent of *node perturbation* (NP) in the network estimates as follows: $NP_j = \sum_i |V_{ij}|$; for FGL we get $\mathbf{V}$ from the PNJGL formulation as $\frac{1}{2}(\hat{\Theta}^1 - \hat{\Theta}^2)$. If $NP_j = 0$ (using a zero-threshold of $10^{-6}$), then the $j$th gene has the same edge weights in the two conditions. In Figure 4(a)-(b), we plotted the resulting values for each of the 258 genes in FGL and PNJGL. Although the network estimates resulting from PNJGL and FGL have approximately the same number of edges that differ across cancer subtypes, PNJGL results in estimates in which only 37 genes appear to have node perturbation. FGL results in estimates in which all 258 genes appear to have node perturbation. In Figure 4(c)-(d), the non-zero elements of $\hat{\Theta}^1 - \hat{\Theta}^2$ for FGL and for PNJGL are displayed. Clearly, the pattern of network differences resulting from PNJGL is far more structured. The genes known to be frequently mutated across GBM subtypes are somewhat enriched out of those that appear to be perturbed according to the PNJGL estimates (3 out of 10 mutated genes were detected by PNJGL; 37 out of 258 total genes were detected by PNJGL; hypergeometric p-value = 0.1594). In contrast, FGL detects *every* gene as having node perturbation (Figure 4(a)). The gene with the highest $NP_j$ value (according to both FGL and PNJGL with $q = 2$) is CXCL13, a small cytokine that belongs to the CXC chemokine family. Together with its receptor CXCR5, it controls the organization of B-cells within follicles of lymphoid tissues. This gene was not identified as a frequently mutated gene in GBM [26]. However, there is recent evidence that CXCL13 plays a critical role in driving cancerous pathways in breast, prostate, and ovarian tissue [27, 28]. Our results suggest the possibility of a previously unknown role of CXCL13 in brain cancer.

## 6 Discussion and future work

We have proposed the *perturbed-node joint graphical lasso*, a new approach for jointly learning Gaussian graphical models under the assumption that network differences result from *node pertur-bations*. We impose this structure using a novel *RCON penalty*, which encourages the differences between the estimated networks to be the union of just a few rows and columns. We solve the resulting convex optimization problem using ADMM, which is more efficient and scalable than standard interior point methods. Our proposed approach leads to far better performance on synthetic data than two alternative approaches: learning Gaussian graphical models assuming *edge perturbation* [13], or simply learning each model separately. Future work will involve other forms of structured sparsity beyond simply node perturbation. For instance, if certain subnetworks are known *a priori* to be related to the conditions under study, then the RCON penalty can be modified in order to encourage some subnetworks to be perturbed across the conditions. In addition, the ADMM algorithm described in this paper requires computation of the eigen decomposition of a $p \times p$ matrix at each iteration; we plan to develop computational improvements that mirror recent results on related problems in order to reduce the computations involved in solving the FGL optimization problem [6, 13].

**Acknowledgments** D.W. was supported by NIH Grant DP5OD009145, M.F. was supported in part by NSF grant ECCS-0847077.

# References

[1] K.V. Mardia, J. Kent, and J.M. Bibby. *Multivariate Analysis*. Academic Press, 1979.

[2] S.L. Lauritzen. *Graphical Models*. Oxford Science Publications, 1996.

[3] M. Yuan and Y. Lin. Model selection and estimation in the Gaussian graphical model. *Biometrika*, 94(10):19–35, 2007.

[4] J. Friedman, T. Hastie, and R. Tibshirani. Sparse inverse covariance estimation with the graphical lasso. *Biostatistics*, 9:432–441, 2007.

[5] O. Banerjee, L. E. El Ghaoui, and A. d'Aspremont. Model selection through sparse maximum likelihood estimation for multivariate Gaussian or binary data. *JMLR*, 9:485–516, 2008.

[6] D.M. Witten, J.H. Friedman, and N. Simon. New insights and faster computations for the graphical lasso. *Journal of Computational and Graphical Statistics*, 20(4):892–900, 2011.

[7] K. Scheinberg, S. Ma, and D. Goldfarb. Sparse inverse covariance selection via alternating linearization methods. *Advances in Neural Information Processing Systems*, 2010.

[8] P. Ravikumar, M.J. Wainwright, G. Raskutti, and B. Yu. Model selection in gaussian graphical models: high-dimensional consistency of l1-regularized MLE. *Advances in NIPS*, 2008.

[9] C.J. Hsieh, M. Sustik, I. Dhillon, and P. Ravikumar. Sparse inverse covariance estimation using quadratic approximation. *Advances in Neural Information Processing Systems*, 2011.

[10] R. Tibshirani. Regression shrinkage and selection via the lasso. *Journal of the Royal Statistical Society, Series B*, 58:267–288, 1996.

[11] A. D'Aspremont, O. Banerjee, and L. El Ghaoui. First-order methods for sparse covariance selection. *SIAM Journal on Matrix Analysis and Applications*, 30(1):56–66, 2008.

[12] J. Guo, E. Levina, G. Michailidis, and J. Zhu. Joint estimation of multiple graphical models. *Biometrika*, 98(1):1–15, 2011.

[13] P. Danaher, P. Wang, and D. Witten. The joint graphical lasso for inverse covariance estimation across multiple classes, 2012. `http://arxiv.org/abs/1111.0324`.

[14] R. Tibshirani, M. Saunders, S. Rosset, J. Zhu, and K. Knight. Sparsity and smoothness via the fused lasso. *Journal of the Royal Statistical Society, Series B*, 67:91–108, 2005.

[15] M. Yuan and Y. Lin. Model selection and estimation in regression with grouped variables. *Journal of the Royal Statistical Society, Series B*, 68:49–67, 2007.

[16] L. Jacob, G. Obozinski, and J.P. Vert. Group lasso with overlap and graph lasso. *Proceedings of the 26th International Conference on Machine Learning*, 2009.

[17] G. Obozinski, L. Jacob, and J.P. Vert. Group lasso with overlaps: the latent group lasso approach. 2011. `http://arxiv.org/abs/1110.0413`.

[18] M. Grant and S. Boyd. `cvx` version 1.21. "http://cvxr.com/cvx", October 2010.

[19] A. Argyriou, C.A. Micchelli, and M. Pontil. Efficient first order methods for linear composite regularizers. 2011. http://arxiv.org/pdf/1104.1436.

[20] X. Chen, Q. Lin, S. Kim, J.G. Carbonell, and E.P. Xing. Smoothing proximal gradient method for general structured sparse learning. *Proceedings of the conference on Uncertainty in Artificial Intelligence*, 2011.

[21] S. Mosci, S. Villa, A. Verri, and L. Rosasco. A primal-dual algorithm for group sparse regularization with overlapping groups. *Neural Information Processing Systems*, pages 2604 – 2612, 2010.

[22] S.P. Boyd, N. Parikh, E. Chu, B. Peleato, and J. Eckstein. Distributed optimization and statistical learning via the alternating direction method of multipliers. *Foundations and Trends in ML*, 3(1):1–122, 2010.

[23] M. Hong and Z. Luo. On the linear convergence of the alternating direction method of multipliers. 2012. Available at `arxiv.org/abs/1208.3922`.

[24] B. He, M. Tao, and X. Yuan. Alternating direction method with gaussian back substitution for separable convex programming. *SIAM Journal of Optimization*, pages 313 – 340, 2012.

[25] J. Duchi and Y. Singer. Efficient online and batch learning using forward backward splitting. *Journal of Machine Learning Research*, pages 2899 – 2934, 2009.

[26] Verhaak et al. Integrated genomic analysis identifies clinically relevant subtypes of glioblastoma characterized by abnormalities in PDGFRA, IDH1, EGFR, and NF1. *Cancer Cell*, 17(1):98–110, 2010.

[27] Grosso et al. Chemokine CXCL13 is overexpressed in the tumour tissue and in the peripheral blood of breast cancer patients. *British Journal Cancer*, 99(6):930–938, 2008.

[28] El-Haibi et al. CXCL13-CXCR5 interactions support prostate cancer cell migration and invasion in a PI3K p110-, SRC- and FAK-dependent fashion. *The Journal of Immunology*, 15(19):5968–73, 2009.

